# Approximate Linear Programming for Average-Cost Dynamic Programming

**Daniela Pucci de Farias**
IBM Almaden Research Center
650 Harry Road, San Jose, CA 95120
pucci@mit.edu

**Benjamin Van Roy**
Department of Management Science and Engineering
Stanford University
Stanford, CA 94305
bvr@stanford.edu

## Abstract

This paper extends our earlier analysis on approximate linear programming as an approach to approximating the cost-to-go function in a discounted-cost dynamic program [6]. In this paper, we consider the average-cost criterion and a version of approximate linear programming that generates approximations to the optimal average cost and differential cost function. We demonstrate that a naive version of approximate linear programming prioritizes approximation of the optimal average cost and that this may not be well-aligned with the objective of deriving a policy with low average cost. For that, the algorithm should aim at producing a good approximation of the differential cost function. We propose a two-phase variant of approximate linear programming that allows for external control of the relative accuracy of the approximation of the differential cost function over different portions of the state space via state-relevance weights. Performance bounds suggest that the new algorithm is compatible with the objective of optimizing performance and provide guidance on appropriate choices for state-relevance weights.

## 1 Introduction

The *curse of dimensionality* prevents application of dynamic programming to most problems of practical interest. Approximate linear programming (ALP) aims to alleviate the curse of dimensionality by approximation of the dynamic programming solution. In [6], we develop a variant of approximate linear programming for the discounted-cost case which is shown to scale well with problem size. In this paper, we extend that analysis to the average-cost criterion.

Originally introduced by Schweitzer and Seidmann [11], approximate linear programming combines the linear programming approach to exact dynamic programming [9] to ap-

proximation of the differential cost function (cost-to-go function, in the discounted-cost case) by a *linear architecture*. More specifically, given a collection of *basis functions* $\phi_i, i = 1, \ldots, K$, mapping states in the system to be controlled to real numbers, approximate linear programming involves solution of a linear program for generating an approximation to the differential cost function of the form $\sum_{i=1}^{K} r_i \phi_i(x)$.

Extension of approximate linear programming to the average-cost setting requires a different algorithm and additional analytical ideas. Specifically, our contribution can be summarized as follows:

**Analysis of the usual formulation of approximate linear programming for average-cost problems.** We start with the observation that the most natural formulation of average-cost ALP, which follows immediately from taking limits in the discounted-cost formulation and can be found, for instance, in [1, 2, 4, 10], can be interpreted as an algorithm for approximating of the optimal average cost. However, to obtain a good policy, one needs a good approximation to the differential cost function. We demonstrate through a counterexample that approximating the average cost and approximating the differential cost function so that it leads to a good policy are not necessarily aligned objectives. Indeed, the algorithm may lead to arbitrarily bad policies, even if the approximate average cost is very close to optimal and the basis functions have the potential to produce an approximate differential cost function leading to a reasonable policy.

**Proposal of a variant of average-cost ALP.** A critical limitation of the average-cost ALP algorithm found in the literature is that it does not allow for external control of how the approximation to the differential cost function should be emphasized over different portions of the state space. In situations like the one described in the previous paragraph, when the algorithm produces a bad policy, there is little one can do to improve the approximation other than selecting new basis functions. To address this issue, we propose a two-phase variant of average-cost ALP: the first phase is simply the average-cost ALP algorithm already found in the literature, which is used for generating an approximation for the optimal average cost. This approximation is used in the second phase of the algorithm for generating an approximation to the differential cost function. We show that the second phase selects an approximate differential cost function minimizing a weighted sum of the distance to the true differential cost function, where the weights (referred to as *state-relevance weights*) are algorithm parameters to be specified during implementation of the algorithm, and can be used to control which states should have more accurate approximations for the differential cost function.

**Development of bounds linking the quality of approximate differential cost functions to the performance of the policy associated with them.** The observation that the usual formulation of ALP may lead to arbitrarily bad policies raises the question of how to design an algorithm for directly optimizing performance of the policy being obtained. With this question in mind, we develop bounds that relate the quality of approximate differential cost functions — i.e., their proximity to the true differential cost function — to the expected increase in cost incurred by using a greedy policy associated with them. The bound suggests using a weighted sum of the distance to the true differential cost function for comparing different approximate differential cost functions. Thus the objective of the second phase of our ALP algorithm is compatible with the objective of optimizing performance of the policy being obtained, and we also have some guidance on appropriate choices of state-relevance weights.

## 2   Stochastic Control Problems and the Curse of Dimensionality

We consider discrete-time stochastic control problems involving a finite state space $\mathcal{S}$ of cardinality $|\mathcal{S}| = N$. For each state $x \in \mathcal{S}$, there is a finite set $\mathcal{A}_x$ of available actions.

When the current state is $x$ and action $a \in \mathcal{A}_x$ is taken, a cost $g_a(x) > 0$ is incurred. State transition probabilities $P_a(x, y)$ represent, for each pair $(x, y)$ of states and each action $a \in \mathcal{A}_x$, the probability that the next state will be $y$ given that the current state is $x$ and the current action is $a \in \mathcal{A}_x$.

A *policy* $u$ is a mapping from states to actions. Given a policy $u$, the dynamics of the system follow a Markov chain with transition probabilities $P_{u(x)}(x, y)$. For each policy $u$, we define a transition matrix $P_u$ whose $(x, y)$th entry is $P_{u(x)}(x, y)$, and a cost vector $g_u$ whose $x$th entry is $g_{u(x)}(x)$. We make the following assumption on the transition probabilities:

**Assumption 1 (Irreducibility).** *For each pair of states $x$ and $y$ and each policy $u$, there is $t$ such that $P_u^t(x, y) > 0$.*

In stochastic control problems, we want to select a policy optimizing a given criterion. In this paper, we will employ as an optimality criterion the average cost $J_u(x) = \lim_{T \to \infty} \frac{1}{T} \mathrm{E}\left[ \sum_{t=0}^{T} g_u(x_t) \Big| x_0 = x \right]$. Irreducibility implies that, for each policy $u$, this limit exists and $J_u(x) = \lambda_u$ for all $x$ — the average cost is independent of the initial state in the system.

We denote the minimal average cost by $\lambda^* \doteq \min_u \lambda_u$. For any policy $u$, we define the associated dynamic programming operator $T_u$ by $T_u h = g_u + P_u h$. Note that $T_u$ operates on vectors $h \in \Re^{|\mathcal{S}|}$ corresponding to functions on the state space $\mathcal{S}$. We also define the dynamic programming operator $T$ by $T h = \min_u T_u h$. A policy $u$ is called *greedy with respect to $h$* if it attains the minimum in the definition of $T$.

An optimal policy minimizing the average cost can be derived from the solution of Bellman's equation $\lambda e + h = T h$, where $e$ is the vector of ones. We denote solutions to Bellman's equation by pairs $(\lambda^*, h^*)$. The scalar $\lambda^*$ is unique and equal to the the optimal average cost. The vector $h^*$ is called a *differential cost function*. The differential cost function is unique up to a constant factor; if $h^*$ solves Bellman's equation, then $h^* + ke$ is also a solution for all $k$, and all other solutions can be shown to be of this form. We can ensure uniqueness by imposing $h^*(x) = 0$ for an arbitrary state $x$. Any policy that is greedy with respect to the differential cost function is optimal.

Solving Bellman's equation involves computing and storing the differential cost function for all states in the system. This is computationally infeasible in most problems of practical interest due to the explosion on the number of states as the number of state variables grows. We try to combat the curse of dimensionality by settling for the more modest goal of finding an approximation to the differential cost function. The underlying assumption is that, in many problems of practical interest, the differential cost function will exhibit some regularity, or structure, allowing for reasonable approximations to be stored compactly.

We consider a linear approximation architecture: given a set of functions $\phi_i : \mathcal{S} \mapsto \Re, i = 1, ...p$, we generate approximations of the form

$$h^*(x) \approx \tilde{h}(x, r) = \sum_{i=1}^{p} r_i \phi_i(x). \tag{1}$$

We define a matrix $\Phi \in \Re^{|\mathcal{S}| \times p}$ by $\Phi = [\ \phi_1\ \cdots\ \phi_p\ ]$, i.e., each of the basis functions is stored as a column of $\Phi$, and each row corresponds to a vector $\phi(x)$ of the basis functions evaluated at a distinct state $x$. We represent $\tilde{h}(\cdot, r)$ in matrix notation as $\Phi r$.

In the remainder of the paper, we assume that (a manageable number of) basis functions are prespecified, and address the problem of choosing a suitable parameter vector $r$. For simplicity, we choose an arbitrary state — henceforth called state "0"— for which we set $h^*(0) = 0$; accordingly, we assume that the basis functions are such that $\phi_i(0) = 0, \forall i$.

## 3  Approximate Linear Programming

Approximate linear programming [11, 6] is inspired by the traditional linear programming approach to dynamic programming, introduced by [9]. Bellman's equation can be solved by the *average-cost exact LP (ELP)*:

$$\max_{\lambda, h} \quad \lambda \tag{2}$$
$$\text{s.t.} \quad \lambda e + h \geq Th.$$

Note that the constraints $\lambda e + h \geq Th$ can be replaced by $\lambda + h(x) \geq g_a(x) + \sum_y P_a(x, y) h(y), \forall x, a$, therefore we can think of problem (2) as an LP.

In approximate linear programming, we reduce the generally intractable dimensions of the average-cost ELP by constraining $h$ to be of the form $\Phi r$. This yields the *first-phase approximate LP (ALP)*

$$\max_{\lambda, r} \quad \lambda \tag{3}$$
$$\text{s.t.} \quad \lambda e + \Phi r \geq T\Phi r.$$

Problem (3) can be expressed as an LP by the same argument used for the exact LP. We denote its solution by $(\lambda_1, r_1)$

The following result is immediate.

**Lemma 1.** *The solution $\lambda_1$ of the first-phase ALP minimizes $|\lambda^* - \lambda|$ over the feasible region.*

**Proof:** Maximizing $\lambda$ in (3) is equivalent to maimizing $\lambda^* - \lambda$. Since the first-phase ALP corresponds to the exact LP (2) with extra constraints $h = \Phi r$, we have $\lambda \leq \lambda^*$ for all feasible $\lambda$. Hence $\lambda^* - \lambda = |\lambda^* - \lambda|$, and the claim follows. □

Lemma 1 implies that the first-phase ALP can be seen as an algorithm for approximating the optimal average cost. Using this algorithm for generating a policy for the average-cost problem is based on the hope that approximation of the optimal average cost should also implicitly imply approximation of the differential cost function. Note that it is not unreasonable to expect that some approximation of the differential cost function should be involved in the minimization of $|\lambda^* - \lambda|$; for instance, we know that $\lambda_1 = \lambda^*$ iff $\Phi r_1 = h^*$.

The ALP has as many variables as the number of basis functions plus one, which will usually amount to a dramatically smaller number of variables than what we had in the ELP. However, the ALP still has as many constraints as the number of state-action pairs. This problem is also found in the discounted-cost formulation and there are several approaches in the literature for dealing with it, including constraint sampling [7] and exploitation of problem-specific structures for efficient elimination of redundant constraints [8, 10].

Our first step in the analysis of average-cost ALP is to demonstrate through a counterexample that it can produce arbitrarily bad policies, even if the approximation to the average cost is very accurate.

## 4  Performance of the first-phase ALP: a counterexample

We consider a Markov process with states $0, 1, ..., B$, each representing a possible number of jobs in a queue with buffer of size $B$. The system state $X_t$ evolves according to

$$X_{t+1} = \begin{cases} X_t - 1, & \text{with probability } q(X_t), \\ X_t + 1, & \text{with probability } p, \\ X_t, & \text{otherwise.} \end{cases}$$

From state 0, transitions to states 1 and 0 occurs with probabilities $p$ and $1 - p$, respectively. From state $B$, transitions to states $B - 1$ and $B$ occur with probabilities $q(B - 1)$ and $1 - q(B - 1)$, respectively. The arrival probability $p$ is the same for all states and we let $p = 0.35$. The action to be chosen in each state $x$ is the departure probability or service rate $q(x)$, which takes values the set $\{0.1625, 0.325, 0.4875, 0.65\}$. The cost incurred at state $x$ if action $q$ is taken is given by $g(x, q) = x^2 + 500q^2$.

We use basis functions $\phi_0(x) = 1 - 1_0(x)$, $\phi_i(x) = x^i, i = 1, \ldots, 3$. For $B \geq 100$, the first-phase ALP yields an approximation $\lambda_1 = 90.3$ for the optimal average cost, which is within 2% of the true value $\lambda^* = 92.0$. However, the average cost yielded by the greedy policy with respect to $\Phi r_1$ is 9842.2 for $B = 100$, and goes to infinity as we increase the buffer size. Figure 1 explains this behavior. Note that $\Phi \tilde{r}_1$ is a very good approximation for $h^*$ over states $x \leq 20$, and becomes progressive worse as $x$ increases. States $x \leq 20$ correspond to virtually all of the stationary probability under the optimal policy ($P(x \leq 20) \approx 0.99999$), hence it is not surprising that the first-phase ALP yields a very accurate approximation for $\lambda^*$, as other states contribute very little to the optimal average cost. However, fitting the optimal average cost and the differential cost function over states visited often under the optimal policy is not sufficient for getting a good policy. Indeed, $\Phi r_1$ severely underestimates costs in large states, and the greedy policy drives the system to those states, yielding a very large average cost and ultimately making the system unstable, when the buffer size goes to infinity.

It is also troublesome to note that our choice of basis function actually has the potential to lead to a reasonably good policy — indeed, for $r = [-58.7 \ 187.2 \ 29.5 \ 0.3]$, the greedy policy associated with $\Phi r$ has an average cost approximately equal to 96.7, regardless of the buffer size, which is only about 5.1% larger than the optimal average cost. Hence even though the first-phase ALP is being given a relatively good set of basis functions, it is producing a bad approximate differential cost function, which cannot be improved unless different basis functions are selected.

## 5  Two-phase average-cost ALP

A striking difference between the first-phase average-cost ALP and discounted-cost ALP is the presence in the latter of *state relevance weights*. These are algorithm parameters that can be used to control the accuracy of the approximation to the cost-to-go function (the discounted-cost counterpart of the differential cost function) over different portions of the state space and have been shown in [6] to have a first-order impact on the performance of the policy being generated. For instance, in the example described in the previous section, in the discounted-cost formulation one might be able to improve the policy yielded by ALP by choosing state-relevance weights that put more emphasis on states $x \geq 20$. Inspired by this observation, we propose a two-phase algorithm with the characteristic that state-relevance weights are present and can be used to control the quality of the differential cost function approximation. The first phase is simply the first-phase ALP introduced in Section 3, and is used for generating an approximation to the optimal average cost. The second phase consists of solving the *second-phase ALP* for finding approximations to the differential cost function:

$$\max_r \quad c^T \Phi r \tag{4}$$
$$\text{s.t.} \quad (T\Phi r)(x) \geq \lambda_2 + (\Phi r)(x), \ \forall x \neq 0.$$

The state-relevance weights $c > 0$ and $\lambda_2$ are algorithm parameters to be specified by the user and $c^T$ denotes the transpose of $c$. We denote the optimal solution of the second-phase ALP by $r_2$.

We now demonstrate how the state-relevance weights and $\lambda_2$ can be used for controlling the quality of the approximation to the differential cost function. We first define, for any

given $\lambda$, the function $h_\lambda$, given by the unique solution to [3]

$$h(x) = (Th)(x) - \lambda, \ \forall x \neq 0, \ h(0) = 0. \tag{5}$$

If $\lambda$ is our estimate for the optimal average cost, then $h_\lambda$ can be seen as an estimate to the differential cost function $h^*$. Our first result links the difference between $h^*$ and $h_\lambda$ to the difference between $\lambda^*$ and $\lambda$, when $\lambda \leq \lambda^*$. For simplicity of notation, we implicitly drop from all vectors and matrices rows and columns corresponding to state 0, so that, for instance, $h^*$ corresponds to the original vector $h^*$ without the row corresponding to state 0, and $P_{u^*}$ corresponds to the original matrix $P_{u^*}$ without rows and columns corresponding to state 0.

**Lemma 2.** *For all $\lambda$, we have*

$$h_\lambda - h^* \leq (\lambda^* - \lambda)(I - P_{u^*})^{-1}e.$$

**Proof:** Equation (5), satisfied by $h_\lambda$, corresponds to Bellman's equation for the problem of finding the stochastic shortest path to state 0, when costs are given by $g_a(x) - \lambda$ [3]. Hence $h_\lambda$ corresponds to the vector of smallest expected lengths of paths until state 0. It follows that

$$
\begin{aligned}
h_\lambda &\leq (I - P_{u^*})^{-1}(g_u - \lambda e) \\
&= (I - P_{u^*})^{-1}(g_u - \lambda^* e + (\lambda^* - \lambda)e) \\
&= h^* + (\lambda^* - \lambda)(I - P_{u^*})^{-1}e.
\end{aligned}
$$

$\square$

Note that if $\lambda \leq \lambda^*$, we also have $h_\lambda \geq h^*$, and $|h_\lambda - h^*| \leq (\lambda^* - \lambda)(I - P_{u^*})^{-1}e$.

In the following theorem, we show that the second-phase ALP minimizes $\|h_{\lambda_2} - \Phi r\|_{1,c}$ over the feasible region. The weighted $L_1$ norm $\|\cdot\|_{1,\nu}$, which will be used in the remainder of the paper, is defined as $\|h\|_{1,\nu} = \sum_x \nu(x)|h(x)|$, for any $\nu > 0$.

**Theorem 1.** *Let $r_2$ be the optimal solution to the second-phase ALP. Then it minimizes $\|h_{\lambda_2} - \Phi r\|_{1,c}$ over the feasible region of the second-phase ALP.*

**Proof:** Maximizing $c^T \Phi r$ is equivalent to minimizing $c^T(h_{\lambda_2} - \Phi r)$. It is a well-known result that, for all $h$ such that $Th - \lambda_2 e \geq h$, we have $h \leq h_{\lambda_2}$. It follows that $\Phi r \leq h_{\lambda_2}$ over the feasible region of the second-phase ALP, and $\Phi r_2$ minimizes $c^T(h_{\lambda_2} - \Phi r) = c^T|h_{\lambda_2} - \Phi r| = \|h_{\lambda_2} - \Phi r\|_{1,c}$. $\square$

For any fixed choice of $\lambda_2$ satisfying $\lambda_2 \leq \lambda^*$, we have

$$\|h^* - \Phi r_2\|_{1,c} \leq \|h_{\lambda_2} - \Phi r_2\|_{1,c} + (\lambda^* - \lambda_2)c^T(I - P_{u^*})^{-1}e, \tag{6}$$

hence the second-phase ALP minimizes an upper bound on the weighted $L_1$ norm $\|h^* - \Phi r_2\|_{1,c}$ of the error in the differential cost function approximation. Note that state-relevance weights $c$ determine how errors over different portions of the state space are weighted in the decision of which approximate differential cost function to select, and can be used for balancing accuracy of the approximation over different states. In the next section, we will provide performance bounds that tie a certain $L_1$ norm of the difference between $h^*$ and $\Phi r$ to the expect increase in cost incurred by using the greedy policy with respect to $\Phi r$. This demonstrates that the objective optimized by the second-phase ALP is compatible with the objective of optimizing performance of the policy being obtained, and it also provides some insight about appropriate choices of state-relevance weights.

We have not yet specified how to choose $\lambda_2$. An obvious choice is $\lambda_2 = \lambda_1$, since $\lambda_1$ is the estimate for the optimal average cost yielded by the first-phase ALP and it satisfies $\lambda_1 \leq \lambda^*$, so that bound (6) holds. In practice, it may be advantageous to perform a line search

over $\lambda_2$ to optimize performance of the ultimate policy being generated. An important issue is the feasibility of the second-phase ALP will be feasible for a given choice of $\lambda_2$; for $\lambda_2 \leq \lambda_1$, this will always be the case. It can also be shown that, under certain conditions on the basis functions $\Phi r$, the second-phase ALP possesses multiple feasible solutions regardless of the choice of $\lambda_2$.

## 6  A performance bound

In this section, we present a bound on the performance of greedy policies associated with approximate differential cost functions. This bound provide some guidance on appropriate choices for state-relevance weights.

**Theorem 2.** *Let Assumption 1 hold. For all $h$, let $\lambda_h$ and $\pi_h$ denote the average cost and stationary state distribution of the greedy policy associated with $h$. Then, for all $h$ such that $h \leq h^*$, we have $\lambda_h \leq \lambda^* + \|h^* - h\|_{1,\pi_h}$.*

**Proof:** We have $\lambda_h = \pi_h^T g_h = \pi_h^T (g_h + P_h h - h) = \pi_h^T (Th - h)$, where $g_h$ and $P_h$ denote the costs and transition matrix associated with the greedy policy with respect to $h$, and we have used $\pi_h^T P_h = \pi_h^T$ in the first equality. Now if $h \leq h^*$, we have $\lambda_h = \pi_h^T (Th - h) \leq \pi_h^T (Th^* - h) = \pi_h^T (h^* + \lambda^* - h) = \lambda^* + \|h^* - h\|_{1,\pi_h}$. $\qquad\square$

Theorem 2 suggests that one approach to selecting state-relevance weights may be to run the second-phase ALP adaptively, using in each iteration weights corresponding to the stationary state distribution associated with the policy generated by the previous iteration. Alternatively, in some cases it may suffice to use rough guesses about the stationary state distribution of the MDP as choices for the state-relevance weights. We revisit the example from Section 4 to illustrate this idea.

**Example 1.** *Consider applying the second-phase ALP to the controlled queue described in Section 4. We use weights of the form $c(x) = (1 - \rho)\rho^x$. This is similar to what is done in [6] and is motivated by the fact that, if the system runs under a "stabilizing" policy, there are exponential lower and upper bounds to the stationary state distribution [5]. Hence $c(\cdot)$ is a reasonable guess for the shape of the stationary distribution. We also let $\lambda_2 = 0.95\lambda_1$.*

*Figure 1 demonstrates the evolution of $\Phi r_2$ as we increase $\rho$. Note that there is significant improvement in the shape of $\Phi r_2$ relative to $\Phi r_1$. The best policy is obtained for $\rho = 0.9$, and incurs an average cost of approximately $96.7$, regardless of the buffer size. This cost is only about $5\%$ higher than the optimal average cost.*

## 7  Conclusions

We have extended the analysis of ALP to the case of minimization of average costs. We have shown how the ALP version commonly found in the literature may lead to arbitrarily bad policies even if the choice of basis functions is relatively good; the main problem is that this version of the algorithm — the first-phase ALP — prioritizes approximation of the optimal average cost, but does not necessarily yield a good approximation for the differential cost function. We propose a variant of approximate linear programming — the two-phase approximate linear programming method — that explicitly approximates the differential cost function. The main attractive of the algorithm is the presence of state-relevance weights, which can be used for controlling the relative accuracy of the differential cost function approximation over different portions of the state space.

Many open issues must still be addressed. Perhaps most important of all is whether there is an automatic way of choosing state-relevance weights. The performance bound suggest in Theorem 2 suggests an iterative scheme, where the second-phase ALP is run multiple

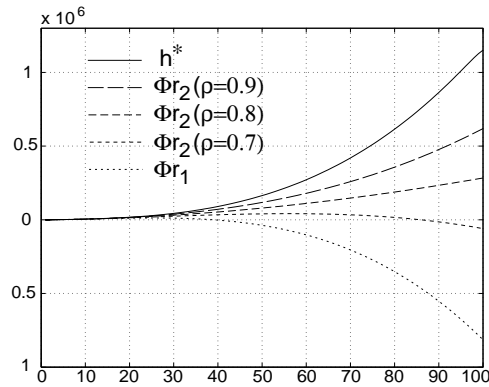

Figure 1: Controlled queue example: Differential cost function approximations as a function of $\rho$. From top to bottom, differential cost function $h^*$, approximations $\Phi r_2$ (with $\rho = 0.9, 0.8, 0.7$), and approximation $\Phi r_1$.

times state-relevance weights are updated in each iteration according to the stationary state distribution obtained with the policy generated by the algorithm in the previous iteration. It remains to be shown whether such a scheme converges. It is also important to note that, in principle, Theorem 2 holds only for $h \leq h^*$. If $\lambda_2 \leq \lambda^*$, this condition cannot be verified for $\Phi r_2$, and the appropriateness of minimizing $\|h^* - \Phi r_2\|_{1,c}$ is only speculative.

# References

[1] D. Adelman. A price-directed approach to stochastic inventory/routing. Preprint, 2002.

[2] D. Adelman. Price-directed replenishment of subsets: Methodology and its application to inventory routing. Preprint, 2002.

[3] D. Bertsekas. *Dynamic Programming and Optimal Control*. Athena Scientific, 1995.

[4] D. Bertsekas and J.N. Tsitsiklis. *Neuro-Dynamic Programming*. Athena Scientific, 1996.

[5] D. Bertsimas, D. Gamarnik, and J.N. Tsitsiklis. Performance of multiclass Markovian queueing networks via piecewise linear Lyapunov functions. *Annals of Applied Probability*, 11.

[6] D.P. de Farias and B. Van Roy. The linear programming approach to approximate dynamic programming. To appear in *Operations Research*, 2001.

[7] D.P. de Farias and B. Van Roy. On constraint sampling in the linear programming approach to approximate dynamic programming. Conditionally accepted to *Mathematics of Operations Research*, 2001.

[8] C. Guestrin, D. Koller, and R. Parr. Efficient solution algorithms for factored MDPs. Submitted to Journal of Artificial Intelligence Research, 2001.

[9] A.S. Manne. Linear programming and sequential decisions. *Management Science*, 6(3):259–267, 1960.

[10] J.R. Morrison and P.R. Kumar. New linear program performance bounds for queueing networks. *Journal of Optimization Theory and Applications*, 100(3):575–597, 1999.

[11] P. Schweitzer and A. Seidmann. Generalized polynomial approximations in Markovian decision processes. *Journal of Mathematical Analysis and Applications*, 110:568–582, 1985.
